# An Auditory Paradigm for Brain–Computer Interfaces

**N. Jeremy Hill**[1], **T. Navin Lal**[1], **Karin Bierig**[1]
**Niels Birbaumer**[2] **and Bernhard Schölkopf**[1]

[1]Max Planck Institute for Biological Cybernetics,
Spemannstraße 38, 72076 Tübingen, Germany.
{jez|navin|bierig|bs}@tuebingen.mpg.de

[2]Institute for Medical Psychology and
Behavioural Neurobiology, University of Tübingen,
Gartenstraße 29, 72074 Tübingen, Germany.
niels.birbaumer@uni-tuebingen.de

## Abstract

Motivated by the particular problems involved in communicating with "locked-in" paralysed patients, we aim to develop a brain-computer interface that uses auditory stimuli. We describe a paradigm that allows a user to make a binary decision by focusing attention on one of two concurrent auditory stimulus sequences. Using Support Vector Machine classification and Recursive Channel Elimination on the independent components of averaged event-related potentials, we show that an untrained user's EEG data can be classified with an encouragingly high level of accuracy. This suggests that it is possible for users to modulate EEG signals in a single trial by the conscious direction of attention, well enough to be useful in BCI.

## 1 Introduction

The aim of research into brain-computer interfaces (BCIs) is to allow a person to control a computer using signals from the brain, without the need for any muscular movement—for example, to allow a completely paralysed patient to communicate. Total or near-total paralysis can result in cases of brain-stem stroke, cerebral palsy, and Amytrophic Lateral Sclerosis (ALS, also known as Lou Gehrig's disease). It has been shown that some patients in a "locked-in" state, in which most cognitive functions are intact despite complete paralysis, can learn to communicate via an interface that interprets electrical signals from the brain, measured externally by electro-encephalogram (EEG) [1]. Successful approaches to such BCIs include using feedback to train the patient to modulate slow cortical potentials (SCPs) to meet a fixed criterion [1], machine classification of signals correlated with imagined muscle movements, recorded from motor and pre-motor cortical areas [2, 3], and detection of an event-related potential (ERP) in response to a visual stimulus event [4].

The experience of clinical groups applying BCI is that different paradigms work to

varying degrees with different patients. For some patients, long immobility and the degeneration of the pyramidal cells of the motor cortex may make it difficult to produce imagined-movement signals. Another concern is that in very severe cases, the entire visual modality becomes unreliable: the eyes cannot adjust focus, the fovea cannot be moved to inspect different locations in the visual scene, meaning that most of a given image will stimulate peripheral regions of retina which have low spatial resolution, and since the responses of retinal ganglion cells that form the input to the visual system are temporally band-pass, complete immobility of the eye means that steady visual signals will quickly fade [5]. Thus, there is considerable motivation to add to the palette of available BCI paradigms by exploring EEG signals that occur in response to auditory stimuli—a patient's sense of hearing is often uncompromised by their condition.

Here, we report the results of an experiment on healthy subjects, designed to develop a BCI paradigm in which a user can make a binary choice. We attempt to classify EEG signals that occur in response to two simultaneous auditory stimulus streams. To communicate a binary decision, the subject focuses attention on one of the two streams, left or right. Hillyard et al. [6] and others reported in the 60's and 70's that selective attention in a dichotic listening task caused a measurable modulation of EEG signals (see [7, 8] for a review). This modulation was significant when signals were averaged over a large number of instances, but our aim is to discover whether *single* trials are classifiable, using machine-learning algorithms, with a high enough accuracy to be useful in a BCI.

## 2  Stimuli and methods

EEG signals were recorded from 15 healthy untrained subjects (9 female, 6 male) between the ages of 20 and 38, using 39 silver chloride electrodes, referenced to the ears. An additional EOG electrode was positioned lateral to and slightly below the left eye, to record eye movement artefacts—blinks and horizontal and vertical saccades all produced clearly identifiable signals on the EOG channel. The signals were filtered by an analog band-pass filter between 0.1 and 40 Hz, before being sampled at 256 Hz.

Subjects sat 1.5m from a computer monitor screen, and performed eight 10-minute blocks each consisting of 50 trials. On each trial, the appearance of a fixation point on screen was followed after 1 sec by an arrow pointing left or right (25 left, 25 right in each block, in random order). The arrow disappeared after 500 msec, after which there was a pause of 500 msec, and then the auditory stimulus was presented, lasting 4 seconds. 500 msec after the end of the auditory stimulus, the fixation point disappeared and there was a pause of between 2 and 4 seconds for the subject to relax. While the fixation point was present, subjects were asked to keep their gaze fixed on it, to blink as little as possible, and not to swallow or make any other movements (we wished to ensure that, as far as possible, our signals were free of artefacts from signals that a paralysed patient would be unable to produce).

The auditory stimulus consisted of two periodic sequences of 50-msec-long square-wave beeps, one presented from a speaker to the left of the subject, and the other from a speaker to the right. Each sequence contained "target" and "non-target" beeps: the first three in the sequence were always non-targets, after which they could be targets with independent probability 0.3. The right-hand sequence consisted of eight beeps of frequencies 1500 Hz (non-target) and 1650 Hz (target), repeating with a period of 490 msec. The left-hand sequence consisted of seven beeps of frequencies 800 Hz (non-target) and 880 Hz (target), starting 70 msec after start of the right-hand sequence and repeating with a period of 555 msec.

According to the direction of the arrow on each trial, subjects were instructed to count the number of target beeps in either the left or right sequence. In the pause between trials, they were instructed to report the number of target beeps using a numeric keypad.[1]

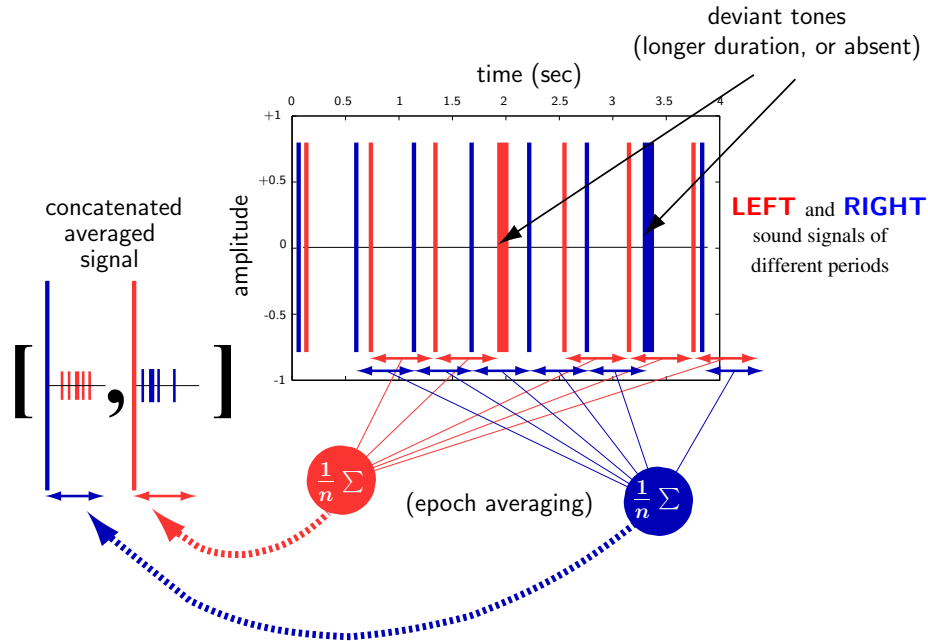

Figure 1: Schematic illustration of the acoustic stimuli used in the experiment, and of the averaging process used in preprocessing method A (illustrated by showing what would happen if the sound signals themselves were averaged)

The sequences differed in location and pitch in order to help the subjects focus their attention on one sequence and ignore the other. The task of reporting the number of target beeps was instituted in order to keep the subjects alert, and to make the task more concrete, because subjects in pilot experiments found that just being asked "listen to the left" or "listen to the right" was too vague a task demand to perform well over the course of 400 repetitions.[2] The regular repetition of the beeps, at the two different periods, was designed to allow the average ERP to a left-hand beep on a single trial to be examined with minimal contamination by ERPs to right-hand beeps, and vice versa: figure 2 illustrates that, when the periods of one signal are averaged, signals correlated with that sequence add in phase, whereas signals correlated with the other sequence spread out, out of phase. Comparison of the average response to a left beat with the average response to a right beat, on a single trial, should thus emphasize any modulating effect of the direction of attention on the ERP, of the kind described by Hillyard et al. [6].

An additional stimulus feature was designed to investigate whether mismatch negativity (MMN) could form a useful basis for a BCI. Mismatch negativity is a difference between the ERP to standard stimuli and the ERP to *deviant* stimuli, i.e. rare stimulus events (with probability of occurrence typically around 0.1) which differ in some manner from the more regular standards. MMN is treated in detail by Näätänen [9]. It has been associated with the distracting effect of the occurrence of a deviant while processing standards, and while it occurs to stimuli outside as well as inside the focus of attention, there is evidence to suggest that this distraction effect is larger the more similar the (task-irrelevant) deviant stimulus is to the (task-relevant) standards [10]. Thus there is the possibility that a deviant stimulus (say, a longer beep) inserted into the sequence to which the subject is attending (same side, same pitch) might elicit a larger MMN signal than a deviant in the unattended sequence. To explore this, after at least two standard beats of each trial, one of the beats (randomly chosen, with the constraint that the epoch following the deviant on the left should not overlap with the epoch following the deviant on the right) was made to deviate on each trial. (Note the frequencies of occurrence of the deviants were 1/7 and 1/8 rather than the ideal 1/10: the double contraint of having manageably short trials and a reasonable epoch length meant that the number of beeps in the left and right sequences was limited to seven and eight respectively, and clearly to use MMN in BCI, every trial has to have at least one deviant in each sequence.) For 8 subjects, the deviant beat was simply a silent beat—a disruptive pause in the otherwise regular sequence. For the remaining 7 subjects, the deviant beat was a beep lasting 100 msec instead of the usual 50 msec (as in the distraction paradigm of Schröger and Wolff [10], the difference between deviant and standard is on a task-irrelevant dimension—in our case duration, the task being to discriminate pitch). A sixteenth subject, in the long-deviant condition, had to be eliminated because of poor signal quality.

## 3  Analysis

As a first step in analyzing the data, the raw EEG signals were examined by eye for each of the 400 trials of each of the subjects. Trials were rejected if they contained obvious large artefact signals caused by blinks or saccades (visible in the EOG and across most of the frontal positions), small periodic eye movements, or other muscle movements (neck and brow, judged from electrode positions O9 and O10, Fp1, Fpz and Fp2). Between 6 and 228 trials had to be rejected out of 400, depending on the subject.

One of two alternative preprocessing methods was then used. In order to look for effects of the attention-modulation reported by Hillyard et al, method (A) took the average ERP in response to standard beats (discarding the first beat). In order to look for possible attention-modulation of MMN, method (B) subtracted the average response to standards from the response to the deviant beat. In both methods, the average ERP signal to beats on the left was concatenated with the average ERP signal following beats on the right, as depicted in figure 2 (for illustrative purposes the figure uses the sound signal itself, rather than an ERP). For each trial, either preprocessing method resulted in a signal of 142 (left) + 125 (right) = 267 time samples for each of 40 channels (39 EEG channels plus one EOG), for a total of 10680 input dimensions to the classifier.

The classifier used was a linear hard-margin Support Vector Machine (SVM) [11]. To evaluate its performance, the trials from a single subject were split into ten non-overlapping partitions of equal size: each such partition was used in turn as a test set for evaluating the performance of the classifier trained on the other 90% of the

trials. Before training, linear Independent Component Analysis (ICA) was carried out on the training set in order to perform blind source separation—this is a common technique in the analysis of EEG data [12, 13], since signals measured through the intervening skull, meninges and cerebro-spinal fluid are of low spatial resolution, and the activity measured from neighbouring EEG electrodes can be assumed to be highly correlated mixtures of the underlying sources. For the purposes of the ICA, the concatenation of all the preprocessed signals from one EEG channel, from all trials in the training partition, was treated as a single mixture signal. A 40-by-40 separating matrix was obtained using the stabilized deflation algorithm from version 2.1 of FastICA [14]. This matrix, computed only from the training set, was then used to separate the signals in both the training set and the test set. Then, the signals were centered and normalized: for each averaged (unmixed) ERP in each of the 40 ICs of each trial, the mean was subtracted, and the signal was divided by its 2-norm. Thus the entry $K_{ij}$ in the kernel matrix of the SVM was proportional to the sum of the coefficients of correlation between corresponding epochs in trials $i$ and $j$. The SVM was then trained and tested. Single-trial error rate was estimated as the mean proportion of misclassified test trials across the ten folds. For comparison, the classification was also performed on the mixture signals without ICA, and with and without the normalizing step.

Results are shown in table 1. Due to space constraints, standard error values for the estimated error rates are not shown: standard error was typically $\pm 0.025$, and maximally $\pm 0.04$. It can be seen that the best error rate obtainable with a given subject varies according to the subject, between 3% and 37%, in a way that is not entirely explained by the differences in the numbers of good (artefact-free) trials available. ICA generally improved the results, by anything up to 14%. Preprocessing method (B) generally performed poorly (minimum 19% error, and generally over 35%). Any attention-dependent modulation of an MMN signal is apparently too small relative to the noise (signals from method B were generally noisier than those from method A, because the latter, averaged over 5 or 6 epochs within a trial, are subtracted from signals that come from only one epoch per trial in order to produce the method B average). For preprocessing method A, normalization generally produced a small improvement.

Thus, promising results can be obtained using the average ERP in response to standard beeps, using ICA followed by normalization (fourth results column): error rates of 5–15% for some some subjects are comparable with the performance of, for example, well-trained patients in an SCP paradigm [1], and correspond to information transfer rates of 0.4–0.7 bits per trial (say, 4–7 bits per minute). Note that, despite the fact that this method does not use the ERPs that occur in response to deviant beats, the results for subject in the silent-deviant condition were generally better than for those in the long-deviant condition. It may be that the more irregular-sounding sequences with silent beats forced the subjects to concentrate harder in order to perform the counting task—alternatively, it may simply be that this group of subjects could concentrate less well, an interpretation which is also suggested by the fact that more trials had to be rejected from their data sets).

In order to examine the extent to which the dimensionality of the classification problem could be reduced, recursive feature elimination [15] was performed (limited now to preprocessing method A with ICA and normalization). For each of ten folds, ICA and normalization was performed, then an SVM was trained and tested. For each independent component $j$, an elimination criterion value $c_j = \sum_{i \in F_j} w_i^2$ was computed, where $\mathbf{w}$ is the hyperplane normal vector of the trained SVM, and $F_j$ is the set of indices to features that are part of component $j$. The IC with the lowest criterion score $c_j$ was deemed to be the least influential for classification,

Table 1: SVM classification error rates: the best rates for each of the preprocessing methods, A and B (see text), are in **bold**. The symbol $\|\cdot\|$ is used to denote normalization during pre-processing as described in the text, and the symbol $\cdot$ is used to denote no normalization.

| subj. | deviant duration (msec) | # good trials | Method A | | | | Method B | | | |
|---|---|---|---|---|---|---|---|---|---|---|
| | | | no ICA | | ICA | | no ICA | | ICA | |
| | | | $\cdot$ | $\|\cdot\|$ | $\cdot$ | $\|\cdot\|$ | $\cdot$ | $\|\cdot\|$ | $\cdot$ | $\|\cdot\|$ |
| CM | 0 | 326 | 0.08 | 0.06 | 0.06 | **0.04** | 0.36 | 0.35 | 0.26 | **0.25** |
| CN | 0 | 250 | 0.26 | 0.19 | 0.28 | **0.14** | 0.43 | 0.44 | **0.38** | 0.40 |
| GH | 0 | 198 | 0.34 | 0.27 | 0.35 | **0.22** | 0.41 | 0.41 | **0.39** | 0.43 |
| JH | 0 | 348 | 0.21 | 0.19 | 0.14 | **0.08** | 0.31 | 0.42 | **0.28** | 0.35 |
| KT | 0 | 380 | 0.23 | 0.21 | 0.15 | **0.07** | 0.41 | 0.36 | 0.35 | **0.34** |
| KW | 0 | 394 | 0.18 | 0.14 | 0.06 | **0.03** | 0.34 | 0.39 | **0.19** | 0.23 |
| TD | 0 | 371 | 0.22 | 0.18 | 0.15 | **0.10** | 0.35 | 0.39 | 0.29 | **0.28** |
| TT | 0 | 367 | 0.32 | **0.31** | 0.33 | 0.32 | 0.40 | 0.42 | **0.39** | 0.43 |
| AH | 100 | 353 | 0.22 | 0.22 | 0.17 | **0.16** | 0.41 | **0.41** | 0.45 | 0.46 |
| AK | 100 | 172 | 0.35 | 0.31 | 0.34 | **0.22** | 0.50 | 0.46 | 0.50 | **0.42** |
| CG | 100 | 271 | 0.37 | 0.29 | 0.31 | **0.28** | 0.51 | 0.47 | 0.48 | **0.44** |
| CH | 100 | 375 | 0.31 | 0.28 | 0.26 | **0.22** | 0.49 | 0.46 | 0.46 | **0.44** |
| DK | 100 | 241 | 0.34 | 0.34 | 0.35 | **0.30** | 0.45 | 0.44 | 0.42 | **0.40** |
| KB | 100 | 363 | 0.21 | 0.21 | 0.15 | **0.10** | 0.42 | 0.47 | **0.39** | 0.41 |
| SK | 100 | 239 | 0.47 | 0.43 | 0.40 | **0.37** | 0.46 | 0.49 | **0.45** | 0.51 |

and the corresponding features $F_j$ were removed. Then the SVM was re-trained and re-tested, and the elimination process iterated until one channel remained. The removal of batches of features in this way is similar to the Recursive Channel Elimination approach to BCI introduced by Lal et al. [3], except that independent components are removed instead of mixtures (a convenient acronym would therefore be RICE, for Recursive Independent Component Elimination).

Results for the two subject groups are plotted in the left and right panels of figure 3, showing estimated error rates averaged over ten folds against the number of ICs used for classification. Each subject's initials, together with the number of useable trials that subject performed, are printed to the right of the corresponding curve.[3] It can be seen that a fairly large number of ICs (around 20–25 out of the 40) contribute to the classification: this may indicate that the useful information in the EEG signals is diffused fairly widely between the areas of the brain from which we are detecting signals (indeed, this is in accordance with much auditory-ERP and -MMN research, in which strong signals are often measured at the vertex, quite far from the auditory cortex [6, 7, 8, 9]). One of the motivations for reducing the dimensionality of the data is to determine whether performance can be improved as irrelevant noise is eliminated, and as the probability of overfitting decreases. However, these factors do not seem to limit performance on the current data: for most subjects, performance does not improve as features are eliminated, instead remaining roughly constant until fewer than 20–25 ICs remain. A possible exception is KT, whose performance may improve by 2–3% after elimination of 20 components, and a clearer exception

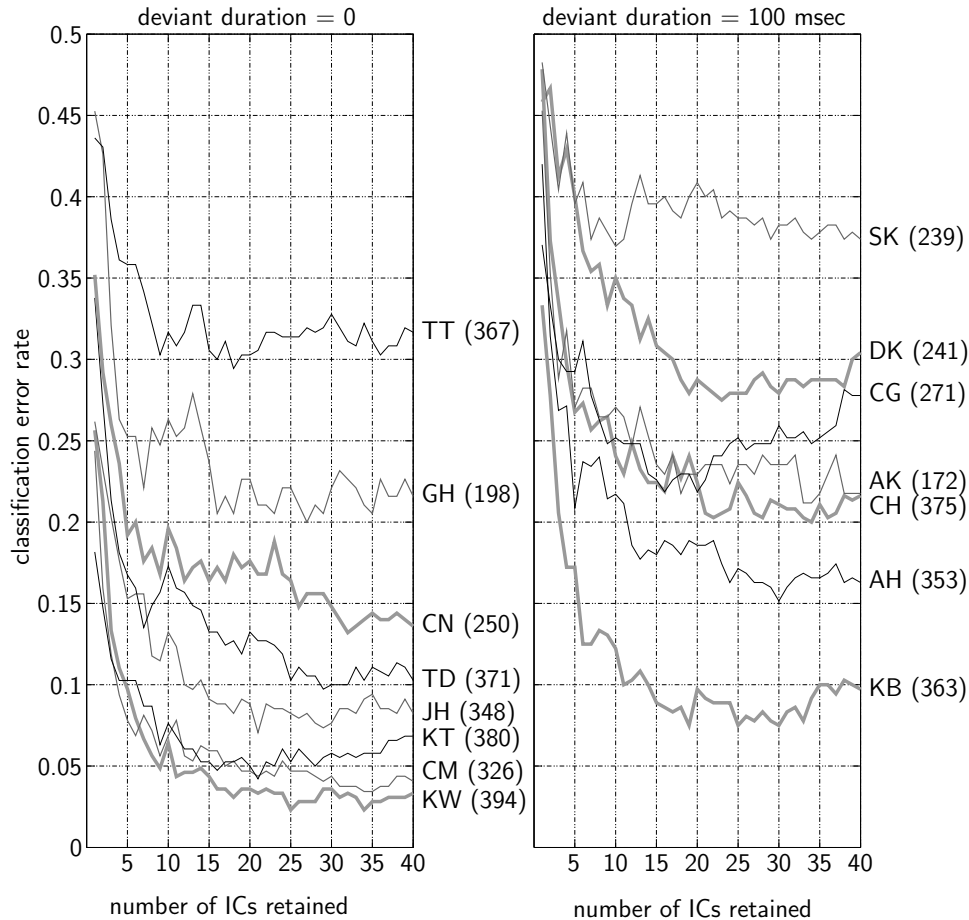

Figure 2: Results of recursive independent component elimination

is CG, for whom elimination of 25 components yields an improvement of roughly 10%.

The ranking returned by the RICE method is somewhat difficult to interpret, not least because each fold of the procedure can compute a different ICA decomposition, whose independent components are not necessarily readily identifiable with one another. A thorough analysis is not possible here—however, with the mixture weightings for many ICs spread very widely around the electrode array, we found no strong evidence for or against the particular involvement of muscle movement artefact signals in the classification.

## 4   Conclusion

Despite wide variation in performance between subjects, which is to be expected in the analysis of EEG data, our classification results suggest that it is possible for a user with no previous training to direct conscious attention, and thereby modulate the event-related potentials that occur in response to auditory stimuli reliably enough, on a single trial, to provide a useful basis for a BCI. The information used by the classifier seems to be diffused fairly widely over the scalp. While

the ranking from recursive independent component elimination did not reveal any evidence of an overwhelming contribution from artefacts related to muscle activity, it is not possible to rule out completely the involvement of such artefacts—possibly the only way to be sure of this is to implement the interface with locked-in patients, preparations for which are underway.

## Acknowledgments

Many thanks to Prof. Kuno Kirschfeld and Bernd Battes for the use of their laboratory.

## Footnotes

[1]In order to avoid contamination of the EEG signals with movement artefacts, a few practice trials were performed before the first block, so that subjects learned to wait until the fixation point was out before looking at the keypad or beginning the hand movement toward it.

[2]Although a paralysed patient would clearly be unable to give responses in this way, it is hoped that this extra motivation would not be necessary.

[3]RICE was also carried out using the full 400 trials for each subject (results not shown). Despite the (sometimes drastic) reduction in the number of trials, rejection by eye of artefact trials did not raise the classification error rate by an appreciable amount. The one exception was subject SK, for whom the probability of mis-classification increased by about 0.1 when 161 trials containing strong movement signals were removed—clearly this subject's movements were classifiably dependent on whether he was attending to the left or to the right.

## References

[1] N. Birbaumer, A. Kübler, N. Ghanayim, T. Hinterberger, J. Perelmouter, J. Kaiser, I. Iversen, B. Kotchoubey, N. Neumann, and H. Flor. The Thought Translation Device (TTD) for Completely Paralyzed Patients. *IEEE Transactions on Rehabilitation Engineering*, 8(2):190–193, June 2000.

[2] G. Pfurtscheller., C. Neuper amd A. Schlögl, and K. Lugger. Separability of EEG signals recorded during right and left motor imagery using adaptive autoregressive parameters. *IEEE Transactions on Rehabilitation Engineering*, 6(3):316–325, 1998.

[3] T.N. Lal, M. Schröder, T. Hinterberger, J. Weston, M. Bogdan, N. Birbaumer, and B. Schölkopf. Support Vector Channel Selection in BCI. *IEEE Transactions on Biomedical Engineering. Special Issue on Brain-Computer Interfaces*, 51(6):1003–1010, June 2004.

[4] E. Donchin, K.M. Spencer, and R. Wijesinghe. The menatal prosthesis: Assessing the speed of a P300-based brain-computer interface. *IEEE Transactions on Rehabilitation Engineering*, 8:174–179, 2000.

[5] L.A. Riggs, F. Ratliff, J.C. Cornsweet, and T.N. Cornsweet. The disappearance of steadily fixated visual test objects. *Journal of the Optical Society of America*, 43:495–501, 1953.

[6] S.A. Hillyard, R.F. Hink, V.L. Schwent, and T.W. Picton. Electrical signs of selective attention in the human brain. *Science*, 182:177–180, 1973.

[7] R. Näätänen. Processing negativity: an evoked-potential reflection of selective attention. *Psychological Bulletin*, 92(3):605–640, 1982.

[8] R. Näätänen. The role of attention in auditory information processing as revealed by event-related potentials and other brain measures of cognitive function. *Behavioral and Brain Sciences*, 13:201–288, 1990.

[9] R. Näätänen. *Attention and Brain Function*. Erlbaum, Hillsdale NJ, 1992.

[10] E. Schröger and C. Wolff. Behavioral and electrophysiological effects of task-irrelevant sound change: a new distraction paradigm. *Cognitive Brain Research*, 7:71–87, 1998.

[11] B. Schölkopf and A. Smola. *Learning with Kernels*. MIT Press, Cambridge, USA, 2002.

[12] K.R. Müller, J. Kohlmorgen, A. Ziehe, and B. Blankertz. Decomposition algorithms for analysing brain signals. In S. Haykin, editor, *Adaptive Systems for Signal Processing, Communications and Control*, pages 105–110, 2000.

[13] A. Delorme and S. Makeig. EEGLAB: an open source toolbox for analysis of single-trial EEG dynamics including Independent Component Analysis. *Journal of Neuroscience Methods*, 134:9–21, 2004.

[14] A. Hyvärinen. Fast and robust fixed-point algorithms for Independent Component Analysis. *IEEE Transactions on Neural Networks*, 10(3):626–634, 1999.

[15] I. Guyon, J. Weston, S. Barnhill, and V. Vapnik. Gene Selection for Cancer Classification using Support Vector Machines. *Journal of Machine Learning Research*, 3:1439–1461, 2003.
